# How Prior Probability Influences Decision Making: A Unifying Probabilistic Model

**Yanping Huang**
University of Washington
huangyp@cs.washington.edu

**Abram L. Friesen**
University of Washington
afriesen@cs.washington.edu

**Timothy D. Hanks**
Princeton University
thanks@princeton.edu

**Michael N. Shadlen**
Columbia University
Howard Hughes Medical Institute
ms4497@columbia.edu

**Rajesh P. N. Rao**
University of Washington
rao@cs.washington.edu

## Abstract

How does the brain combine prior knowledge with sensory evidence when making decisions under uncertainty? Two competing descriptive models have been proposed based on experimental data. The first posits an additive offset to a decision variable, implying a static effect of the prior. However, this model is inconsistent with recent data from a motion discrimination task involving temporal integration of uncertain sensory evidence. To explain this data, a second model has been proposed which assumes a time-varying influence of the prior. Here we present a normative model of decision making that incorporates prior knowledge in a principled way. We show that the additive offset model and the time-varying prior model emerge naturally when decision making is viewed within the framework of partially observable Markov decision processes (POMDPs). Decision making in the model reduces to (1) computing beliefs given observations and prior information in a Bayesian manner, and (2) selecting actions based on these beliefs to maximize the expected sum of future rewards. We show that the model can explain both data previously explained using the additive offset model as well as more recent data on the time-varying influence of prior knowledge on decision making.

## 1 Introduction

A fundamental challenge faced by the brain is to combine noisy sensory information with prior knowledge in order to perceive and act in the natural world. It has been suggested (e.g., [1, 2, 3, 4]) that the brain may solve this problem by implementing an approximate form of Bayesian inference. These models however leave open the question of how actions are chosen given probabilistic representations of hidden state obtained through Bayesian inference. Daw and Dayan [5, 6] were among the first to study decision theoretic and reinforcement learning models with the goal of interpreting results from various neurobiological experiments. Bogacz and colleagues proposed a model that combines a traditional decision making model with reinforcement learning [7] (see also [8, 9]).

In the decision making literature, two apparently contradictory models have been suggested to explain how the brain utilizes prior knowledge in decision making: (1) a model that adds an offset to a

decision variable, implying a static effect of changes to the prior probability [10, 11, 12], and (2) a model that adds a time varying weight to the decision variable, representing the changing influence of prior probability over time [13]. The LATER model (Linear Approach to Threshold with Ergodic Rate), an instance of the additive offset model, incorporates prior probability as the starting point of a linearly rising decision variable and successfully predicts changes to saccade latency when discriminating between two low contrast stimuli [10]. However, the LATER model fails to explain data from the random dots motion discrimination task [14] in which the agent is presented with noisy, time-varying stimuli and must continually process this data in order to make a correct choice and receive reward. The drift diffusion model (DDM), which uses a random walk accumulation, instead of a linear rise to a boundary, has been successful in explaining behavioral and neurophysiological data in various perceptual discrimination tasks [14, 15, 16]. However, in order to explain behavioral data from recent variants of random dots tasks in which the prior probability of motion direction is manipulated [13], DDMs require the additional assumption of dynamic reweighting of the influence of the prior over time.

Here, we present a normative framework for decision making that incorporates prior knowledge and noisy observations under a reward maximization hypothesis. Our work is inspired by models which cast human and animal decision making in a rational, or optimal, framework. Frazier & Yu [17] used dynamic programming to derive an optimal strategy for two-alternative forced choice tasks under a stochastic deadline. Rao [18] proposed a neural model for decision making based on the framework of partially observable Markov decision processes (POMDPs) [19]; the model focuses on network implementation and learning but assumes a fixed deadline to explain the collapsing decision threshold seen in many decision making tasks. Drugowitsch et al. [9] sought to explain the collapsing decision threshold by combining a traditional drift diffusion model with reward rate maximization; their model also requires knowledge of decision time in hindsight. In this paper, we derive a novel POMDP model from which we compute the optimal behavior for sequential decision making tasks. We demonstrate our model's explanatory power on two such tasks: the random dots motion discrimination task [13] and Carpenter and Williams' saccadic eye movement task [10]. We show that the urgency signal, hypothesized in previous models, emerges naturally as a collapsing decision boundary with no assumption of a decision deadline. Furthermore, our POMDP formulation enables incorporation of partial or incomplete prior knowledge about the environment. By fitting model parameters to the psychometric function in the neutral prior condition (equal prior probability of either direction), our model accurately predicts both the psychometric function and the reaction times for the biased (unequal prior probability) case, without introducing additional free parameters. Finally, the same model also accurately predicts the effect of prior probability changes on the distribution of reaction times in the Carpenter and Williams task, data that was previously interpreted in terms of the additive offset model.

## 2 Decision Making in a POMDP framework

### 2.1 Model Setup

We model a decision making task using a POMDP, which assumes that at any particular time step, $t$, the environment is in a particular *hidden* state, $x \in \mathcal{X}$, that is not directly observable by the animal. The animal can make sensory measurements in order to observe noisy samples of this hidden state. At each time step, the animal receives an observation (stimulus), $s_t$, from the environment as determined by an *emission* distribution, $\Pr(s_t|x)$. The animal must maintain a *belief* over the set of possible true world states, given the observations it has made so far: $b_t(x) = \Pr(x|s_{1:t})$, where $s_{1:t}$ represents the sequence of stimuli that the animal has received so far, and $b_0(x)$ represents the animal's prior knowledge about the environment. At each time step, the animal chooses an action, $a \in \mathcal{A}$ and receives an observation and a reward, $R(x, a)$, from the environment, depending on the current state and the action taken. The animal uses Bayes rule to update its belief about the environment after each observation. Through these interactions, the animal learns a policy, $\pi(b) \in \mathcal{A}$ for all $b$, which dictates the action to take for each belief state. The goal is to find an optimal policy, $\pi^*(b)$, that maximizes the animal's total expected future reward in the task.

For example, in the random dots motion discrimination task, the hidden state, $x$, is composed of both the coherence of the random dots $c \in [0, 1]$ and the direction $d \in \{-1, 1\}$ (corresponding to leftward and rightward motion, respectively), neither of which are known to the animal. The

animal is shown a movie of randomly moving dots, a fraction of which are moving in the same direction (this fraction is the coherence). The movie is modeled as a sequence of time varying stimuli $s_{1:t}$. Each frame, $s_t$, is a snapshot of the changes in dot positions, sampled from the emission distribution $s_t \sim \Pr(s_t|kc, d)$, where $k > 0$ is a free parameter that determines the scale of $s_t$. In order to discriminate the direction given the stimuli, the animal uses Bayes rule to compute the posterior probability of the static joint hidden state, $\Pr(x = kdc|s_{1:t})^1$. At each time step, the animal chooses one of three actions, $a \in \{A_R, A_L, A_S\}$, denoting rightward eye movement, leftward eye movement, and sampling (i.e., waiting for one more observation), respectively. When the animal makes a correct choice (i.e., a rightward eye movement $a = A_R$ when $x > 0$ or a leftward eye movement $a = A_L$ when $x < 0$), the animal receives a positive reward $R_P > 0$. The animal receives a negative reward (penalty) or no reward when an incorrect action is chosen, $R_N \leq 0$. We assume that the animal is motivated by hunger or thirst to make a decision as quickly as possible and model this with a unit penalty $R_S = -1$, representing the cost the agent needs to pay when choosing the sampling action $A_S$.

## 2.2 Bayesian Inference of Hidden State from Prior Information and Noisy Observations

In a POMDP, decisions are made based on the belief state $b_t(x) = \Pr(x|s_{1:t})$, which is the posterior probability distribution over $x$ given a sequence of observations $s_{1:t}$. The initial belief $b_0(x)$ represents the animal's prior knowledge about $x$. In both the Carpenter and William's task [10] and the random dots motion discrimination task [13], prior information about the probability of a specific direction (we use rightward direction here, $d_R$, without loss of generality) is learned by the subjects, $\Pr(d_R) = \Pr(d = 1) = \Pr(x > 0) = 1 - \Pr(d_L)$. Consider the random dots motion discrimination task. Unlike the traditional case where a full prior distribution is given, this direction-only prior information provides only partial knowledge about the hidden state which also includes coherence. In the least informative case, only $\Pr(d_R)$ is known and we model the distribution over the remaining components of $x$ as a uniform distribution. Combining this with the direction prior, which is Bernoulli distributed, gives a piecewise uniform distribution for the prior, $b_0(x)$. In the general case, we can express the distribution over coherence as a normal distribution parameterized by $\mu_0$ and $\sigma_0$, resulting in a piecewise normal prior over $x$,

$$b_0(x) = Z_0^{-1} \mathcal{N}(x \mid \mu_0, \sigma_0) \times \begin{cases} \Pr(d_R) & x \geq 0 \\ \Pr(d_L) & x < 0, \end{cases} \tag{1}$$

where $Z_t = \Pr(d_R)(1 - \Phi(0 \mid \mu_t, \sigma_t)) + \Pr(d_L)\Phi(0 \mid \mu_t, \sigma_t)$ is the normalization factor and $\Phi(x \mid \mu, \sigma) = \int_{-\infty}^{x} \mathcal{N}(x \mid \mu, \sigma)dx$ is the cumulative distribution function (CDF) of the normal distribution. The piecewise uniform prior is then just a special case with $\mu_0 = 0$ and $\sigma_0 = \infty$.

We assume the emission distribution is also normally-distributed, $\Pr(s_t|x) = \mathcal{N}(s_t|x, \sigma_e^2)$, which, from Bayes' rule, results in a piecewise normal posterior distribution

$$b_t(x) = Z_t^{-1} \mathcal{N}(x \mid \mu_t, \sigma_t) \times \begin{cases} \Pr(d_R) & x \geq 0 \\ \Pr(d_L) & x < 0 \end{cases} \tag{2}$$

$$\text{where} \quad \mu_t = \left(\frac{\mu_0}{\sigma_0^2} + \frac{t\bar{s}_t}{\sigma_e^2}\right) / \left(\frac{1}{\sigma_0^2} + \frac{t}{\sigma_e^2}\right), \tag{3}$$

$$\sigma_t^2 = \left(\frac{1}{\sigma_0^2} + \frac{t}{\sigma_e^2}\right)^{-1}, \tag{4}$$

and the running average $\bar{s}_t = \sum_{t'=1}^{t} s_{t'}/t$. Consequently, the posterior distribution depends only on $\bar{s}$ and $t$, the two sufficient statistics of the sequence $s_{1:t}$. For the case of a piecewise uniform prior, the variance $\sigma_t^2 = \frac{\sigma_e^2}{t}$, which decreases inversely in proportion to elapsed time. Unless otherwise mentioned, we fix $\sigma_e = 1$, $\sigma_0 = \infty$ and $\mu_0 = 0$ for the rest of this paper because we can rescale the POMDP time step $t' = \frac{t}{\sigma_e}$ to compensate.

## 2.3 Finding the optimal policy by reward maximization

Within the POMDP framework, the animal's goal is to find an optimal *policy* $\pi^*(b_t)$ that maximizes its expected reward, starting at $b_t$. This is encapsulated in the *value function*

$$v^\pi(b_t) = \mathsf{E}\left[\sum_{k=1}^\infty r(b_{t+k}, \pi(b_{t+k})) \mid b_t, \pi\right] \tag{5}$$

where the expectation is taken with respect to all future belief states $(b_{t+1}, \ldots, b_{t+k}, \ldots)$ given that the animal is using $\pi$ to make decisions, and $r(b, a)$ is the reward function over belief states or, equivalently, the expected reward over hidden states, $r(b, a) = \int_x R(x, a)b(x)dx$. Given the value function, the optimal policy is simply $\pi^*(b) = \arg\max_\pi v^\pi(b)$. In this model, the belief $b$ is parameterized by $\bar{s}_t$ and $t$, so the animal only needs to keep track of these instead of encoding the entire posterior distribution $b_t(x)$ explicitly.

In our model, the expected reward $r(b, a) = \int_x R(x, a)b(x)dx$ is

$$r(b, a) = \begin{cases} R_S, & \text{when } a = A_S \\ Z_t^{-1}[\, R_P \Pr(d_R)\,(1 - \Phi(0 \mid \mu_t, \sigma_t)) + R_N \Pr(d_L)\Phi(0 \mid \mu_t, \sigma_t)\,], & \text{when } a = A_R \\ Z_t^{-1}[\, R_N \Pr(d_R)\,(1 - \Phi(0 \mid \mu_t, \sigma_t)) + R_P \Pr(d_L)\Phi(0 \mid \mu_t, \sigma_t)\,], & \text{when } a = A_L \end{cases} \tag{6}$$

where $\mu_t$ and $\sigma_t$ are given by (3) and (4), respectively. The above equations can be interpreted as follows. With probability $\Pr(d_L) \cdot \Phi(0 \mid \mu_t, \sigma_t)$, the hidden state $x$ is less than 0, making $A_R$ an incorrect decision and resulting in a penalty $R_N$ if chosen. Similarly, action $A_R$ is correct with probability $\Pr(d_R) \cdot [1 - \Phi(0 \mid \mu_t, \sigma_t)]$ and earns a reward of $R_P$. The inverse is true for $A_L$. When $A_S$ is selected, the animal simply receives an observation at a cost of $R_S$.

Computing the value function defined in (5) involves an expectation with respect to future belief. Therefore, we need to compute the transition probabilities over belief states, $T(b_{t+1}|b_t, a)$, for each action. When the animal chooses to sample, $a_t = A_S$, the animal's belief distribution at the next time step is computed by marginalizing over all possible observations [19]

$$T(b_{t+1}|b_t, A_S) = \int_s \Pr(b_{t+1}|s, b_t, A_S)\Pr(s|b_t, A_S)ds \tag{7}$$

$$\text{where} \qquad \Pr(b_{t+1} \mid s, b_t, A_S) = \begin{cases} 1 & \text{if } b_{t+1}(x) = \Pr(s|x)b_t(x)/\Pr(s|b_t, A_S), \forall x \\ 0 & \text{otherwise}; \end{cases} \tag{8}$$

$$\text{and} \qquad \Pr(s \mid b_t, A_S) = \int_x \Pr(s|x)\Pr(x|b, a)dx = \mathsf{E}_{x \sim b}[\Pr(s|x)] \tag{9}$$

When choosing $A_S$, the agent does not affect the world state, so, given the current state $b_t$ and an observation $s$, the updated belief $b_{t+1}$ is deterministic and thus $\Pr(b_{t+1} \mid s, b_t, A_S)$ is a delta function, following Bayes' rule. The probability $\Pr(s \mid b_t, A_S)$ can be treated as a normalization factor and is independent of hidden state[2]. Thus, the transition probability function, $T(b_{t+1} \mid b_t, A_S)$, is solely a function of the belief $b_t$ and is a stationary distribution over the belief space.

When the selected action is $A_L$ or $A_R$, the animal stops sampling and makes an eye movement to the left or the right, respectively. To account for these cases, we include a terminal state, $\Gamma$, with zero-reward (i.e., $R(\Gamma, a) = 0, \forall a$), and absorbing behavior, $T(\Gamma|\Gamma, a) = 1, \forall a$. Moreover, whenever the animal chooses $A_L$ or $A_R$, the POMDP immediately transitions into $\Gamma$: $T(\Gamma|b, a \in \{A_L, A_R\}) = 1, \forall b$, indicating the end of a trial.

Given the transition probability between belief states $T(b_{t+1}|b_t, a)$ and the reward function, we can convert our POMDP model into a Markov Decision Process (MDP) over the belief state. Standard dynamic programming techniques (e.g., value iteration [20]) can then be applied to compute the value function in (5). A neurally plausible method for learning the optimal policy by trial and error using temporal difference (TD) learning was suggested in [18]. Here, we derive the optimal policy from first principles and focus on comparisons between our model's predictions and behavioral data.

# 3 Model Predictions

## 3.1 Optimal Policy

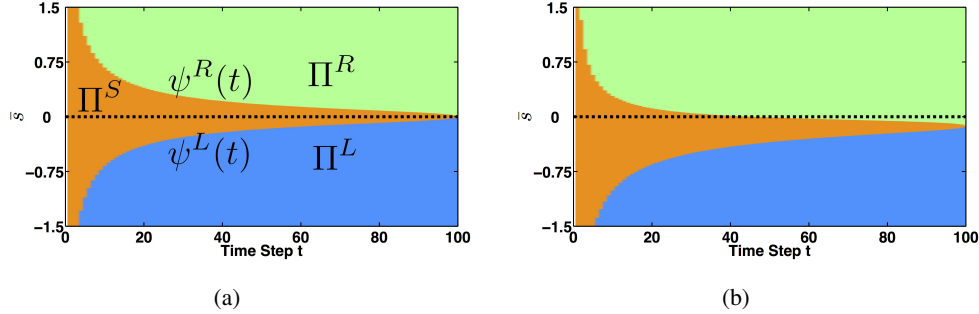

(a)                                                                 (b)

Figure 1: **Optimal policy** for $\Pr(d_R) = 0.5$, and 0.9. (a–b) Optimal policy as a joint function of $\bar{s}$ and $t$. Every point in these figures represents a belief state determined by equations (2), (3) and (4). The color of each point represents the corresponding optimal action. The boundaries $\psi_R(t)$ and $\psi_L(t)$ divide the belief space into three areas $\Pi_S$ (center), $\Pi_R$ (upper) and $\Pi_L$ (lower), respectively. Model parameters: $\frac{R_N - R_P}{R_S} = 1,000$.

Figure 1(a) shows the optimal policy $\pi^*$ as a joint function of $\bar{s}$ and $t$ for the unbiased case where the prior probability $\Pr(d_R) = \Pr(d_L) = 0.5$. $\pi^*$ partitions the belief space into three regions: $\Pi^R$, $\Pi^L$, and $\Pi^S$, representing the set of belief states preferring actions $A_R$, $A_L$ and $A_S$, respectively. We define the boundary between $A_R$ and $A_S$, and the boundary between $A_L$ and $A_S$ as $\psi_R(t)$ and $\psi_L(t)$, respectively. Early in a trial, the model selects the sampling action $A_S$ regardless of the value of the observed evidence. This is because the variance of the running average $\bar{s}$ is high for small $t$. Later in the trial, the model will choose $A_R$ or $A_L$ when $\bar{s}$ is only slightly above 0 because this variance decreases as the model receives more observations. For this reason, the width of $\Pi^S$ diminishes over time. This gradual decrease in the threshold for choosing one of the non-sampling actions $A_R$ or $A_L$ has been called a "collapsing bound" in the decision making literature [21, 17, 22]. For this unbiased prior case, the expected reward function is symmetric, $r(b_t(x), A_R) = r(\Pr(x|\bar{s}_t, t), A_R) = r(\Pr(x|-\bar{s}_t, t), A_L)$, and thus the decision boundaries are also symmetric around 0: $\psi_R(t) = -\psi_L(t)$.

The optimal policy $\pi^*$ is entirely determined by the reward parameters $\{R_P, R_N, R_S\}$ and the prior probability (the standard deviation of the emission distribution $\sigma_e$ only determines the temporal resolution of the POMDP). It applies to both Carpenter and Williams' task and the random dots task (these two tasks differ only in the interpretation of the hidden state $x$). The optimal action at a specific belief state is determined by the relative, not the absolute, value of the expected future reward. From (6), we have

$$r(b, A_L) - r(b, A_R) \propto R_N - R_P. \tag{10}$$

Moreover, if the unit of reward is specified by the sampling penalty, the optimal policy $\pi^*$ is entirely determined by the ratio $\frac{R_N - R_P}{R_S}$ and the prior.

As the prior probability becomes biased, the optimal policy becomes asymmetric. When the prior probability, $\Pr(d_R)$, increases, the decision boundary for the more likely direction ($\psi_R(t)$) shifts towards the center (the dashed line at $\bar{s} = 0$ in figure 1), while the decision boundary for the opposite direction ($\psi_L(t)$) shifts away from the center, as illustrated in Figure 1(b) for prior $\Pr(d_R = 0.9)$. Early in a trial, $\Pi^S$ has approximately the same width as in the neutral prior case, but it is shifted downwards to favor more sampling for $d_L$ ($\bar{s} < 0$). Later in a trial, even for some belief states with $\bar{s} < 0$, the optimal action is still $A_R$, because the effect of the prior is stronger than that of the observed data.

## 3.2 Psychometric function and reaction times in the random dots task

We now construct a decision model from the learned policy for the reaction time version of the motion discrimination task [14], and compare the model's predictions to the psychometric and

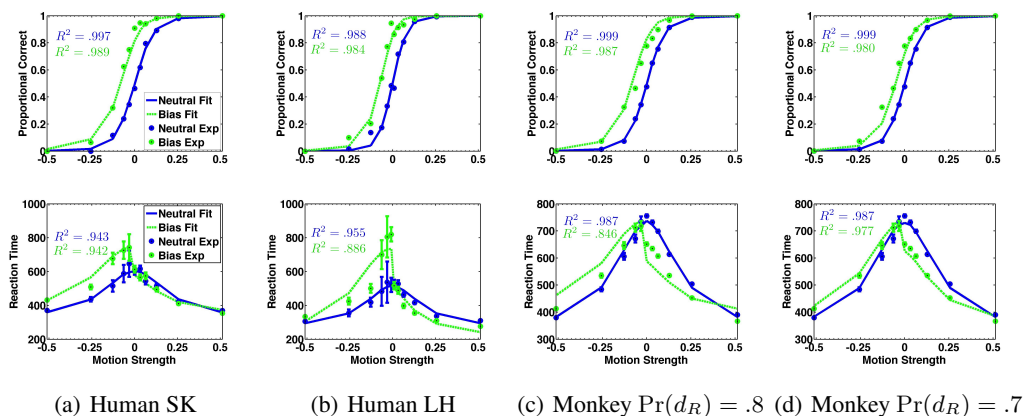

(a) Human SK     (b) Human LH     (c) Monkey $\Pr(d_R) = .8$   (d) Monkey $\Pr(d_R) = .7$

Figure 2: **Comparison of Psychometric (upper panels) and Chronometric (lower panels) functions between the Model and Experiments.** The dots with error bars represent experimental data from human subject SK, and LH, and the combined results from four monkeys. Blue solid curves are model predictions in the neutral case while green dotted curves are model predictions from the biased case. The $R^2$ fits are shown in the plots. Model parameters: (a) $\frac{R_N - R_P}{R_S} = 1,000$, $k = 1.45$. (b) $\frac{R_N - R_P}{R_S} = 1,000$, $\mu = 1.45$. (c) $\Pr(d_R) = 0.8$, $\frac{R_N - R_P}{R_S} = 1,000$, $k = 1.4$. (d) $\Pr(d_R) = 0.7$, $\frac{R_N - R_P}{R_S} = 1,000$, $k = 1.4$.

chronometric functions of a monkey performing the same task [13, 14]. Recall that the belief $b$ is parametrized by $\bar{s}_t$ and $t$, so the animal only needs to know the elapsed time and compute a running average $\bar{s}_t$ of the observations in order to maintain the posterior belief $b_t(x)$. Given its current belief, the animal selects an action from the optimal policy $\pi^*(b_t)$. When $b_t \in \Pi^S$, the animal chooses the sampling action and gets a new observation $s_{t+1}$. Otherwise the animal terminates the trial by making an eye movement to the right or to the left, for $\bar{s}_t > \psi_R(t)$ or $\bar{s}_t < \psi_L(t)$, respectively.

The performance on the task using the optimal policy can be measured in terms of both the accuracy of direction discrimination (the so-called psychometric function), and the reaction time required to reach a decision (the chronometric function). The hidden variable $x = kdc$ encapsulates the unknown direction and coherence, as well as the free parameter $k$ that determines the scale of stimulus $s_t$. Without loss of generality, we fix $d = 1$ (rightward direction), and set the hidden direction $d_R$ as the biased direction. Given an optimal policy, we compute both the psychometric and chronometric function by simulating a large number of trials (10000 trials per data point) and collecting the reaction time and chosen direction from each trial.

The upper panels of figure 2(a) and 2(b) (blue curves) show the performance accuracy as a function of coherence for both the model (blue solid curve) and the human subjects (blue dots) for neutral prior $\Pr(d_R) = 0.5$. We fit our simulation results to the experimental data by adjusting the only two free parameters in our model: $\frac{R_N - R_P}{R_S}$ and $k$. The lower panels of figure 2(a) and 2(b) (blue solid curves) shows the predicted mean reaction time for correct choices as a function of coherence $c$ for our model (blue solid curve, with same model parameters) and the data (blue dots). Note that our model's predicted reaction times represent the expected number of POMDP time steps before making a rightward eye movement $A_R$, which we can directly compare to the monkey's experimental data in units of real time. A linear regression is used to determine the duration $\tau$ of a single time step and the onset of decision time $t_{nd}$. This offset, $t_{nd}$, can be naturally interpreted as the non-decision residual time. We applied the experimental mean reaction time reported in [13] with motion coherence $c = 0.032, 0.064, 0.128, 0.256$ and $0.512$ to compute the slope and offset, $\tau$ and $t_{nd}$. Linear regression gives the unit duration per POMDP step as $\tau = 5.74$ms , and the offset $t_{nd} = 314.6$ms, for human SK. For human LH, similar results are obtained with $\tau = 5.20$ms and $t_{nd} = 250.0$ms. Our predicted offsets compare well with the 300ms non-decision time on average reported in the literature [23, 24].

When the human subject is verbally told that the prior probability is $\Pr(d_R) = \Pr(d = 1) = 0.8$, the experimental data is inconsistent with the predictions of the classic drift diffusion model [14] unless an additional assumption of a dynamic bias signal is introduced. In the POMDP model we propose, we predict both the accuracy and reaction times in the biased setting (green curves in figure 2) with the parameters learned in the neutral case, and achieve a good fit (with the coefficients of determination shown in fig. 2) to the experimental data reported by Hanks et al. [13]. Our model predictions for the biased cases are a direct result of the reward maximization component of our framework and require no additional parameter fitting.

Combined behavioral data from four monkeys is shown by the dotted curves in figure 2(c). We fit our model parameters to the psychometric function in the neutral case, with $\tau = 8.20$ms and $t_{nd} = 312.50$ms, and predict both the psychometric function and the reaction times in the biased case. However, our results do not match the monkey data as well as the human data when $\Pr(d_R) = 0.8$. This may be due to the fact that the monkeys cannot receive verbal instructions from the experimenters and must learn an estimate of the prior during training. As a result, the monkeys' estimate of the prior probability might be inaccurate. To test this hypothesis, we simulated our model with $\Pr(d_R) = 0.7$ (see figure 2(d)) and these results fit the experimental data much more accurately (even though the actual probability was $0.8$).

## 3.3  Reaction times in the Carpenter and Williams' task

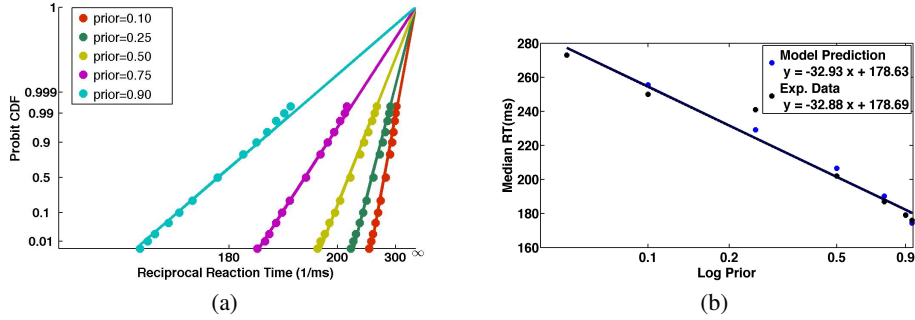

(a)                                                                (b)

Figure 3: **Model predictions of saccadic eye movement in Carpenter & Williams' experiments [10]**. (a) Saccadic latency distributions from model simulations plotted in the form of probit-scale cumulative mass function, as a function of reciprocal latency. For different values of $\Pr(d_R)$, the simulated data are well fit by straight lines, indicating that the reciprocal of latency follows a normal distribution. The solid lines are linear functions fit to the data with the constraint that all lines must pass through the same intercept for infinite time (see [10]). (b) Median latency plotted as a function of log prior probability. Black dots are from experimental data and blue dots are model predictions. The two (overlapping) straight lines are the linear least squares fits to the experimental data and model data. These lines do not differ noticeably in either slope or offset. Model parameters: $\frac{R_N - R_P}{R_S} = 1,000$, $k = 0.3$, $\sigma_e = 0.46$.

In Carpenter and Williams' task, the animal needs to decide on which side $d \in \{-1, 1\}$ (denoting left or right side) a target light appeared at a fixed distance from a central fixation light. After the sudden appearance of the target light, a constant stimulus $s_t = s$ is observed by the animal, where $s$ can be regarded as the perceived location of the target. Due to noise and uncertainty in the nervous system, we assume that $s$ varies from trial to trial, centered at the location of the target light and with standard deviation $\sigma_e$ (i.e., $s \sim \mathcal{N}(s \mid k, \sigma_e^2)$), where $k$ is the distance between the target and the fixation light. Inference over the direction $d$ thus involves joint inference over $(d, k)$ where the emission probability follows $\Pr(s|d, k)$. Then the joint state $(k, d)$ can be one-on-one-mapped to $kd = x$, where $x$ represents the actual location of the target light. Under the POMDP framework, Carpenter and Williams' task and the random dots task differ in the interpretation of hidden state $x$ and stimulus $s$, but they follow the same optimal policy given the same reward parameters.

Without loss of generality, we set the hidden variable $x > 0$ and say that the animal makes a correct choice at a hitting time $t^H$ when the animal's belief state reaches the right boundary. The

saccadic latency can be computed by inverting the boundary function $\psi_R^{-1}(s) = t^H$. Since, for small $t$, $\psi_R(t)$ behaves like a simple reciprocal function of $t$, the reciprocal of the reaction time is approximately proportional to a normal distribution with $\frac{1}{t^H} \sim \mathcal{N}(1/t^H \mid k, \sigma_e^2)$. In figure 3(a), we plot the distribution of reciprocal reaction time with different values of $\Pr(d_R)$ on a probit scale (similar to [10]). Note that we label the $y$-axis using the CDF of the corresponding probit value and the $x$-axis in figure 3(a) has been reversed. If the reciprocal of reaction time (with the same prior $\Pr(d_R)$) follows a normal distribution, each point on the graph will fall on a straight line with $y$-intercept $\frac{k\sqrt{2}}{\sigma_e}$ that is independent of $\Pr(d_R)$. We fit straight lines to the points on the graph, with the constraint that all lines should pass through the same intercept for infinite time (see [10]). We obtain an intercept of $6.19$, consistent with the intercept $6.20$ obtained from experimental data in [10]. Figure 3(b) demonstrates that the median of our model's reaction times is a linear function of the log of the prior probability. Increasing the prior probability lowers the decision boundary $\psi_R(t)$, effectively decreasing the latency. The slope and intercept of the best fit line are consistent with experimental data (see fig. 3(b)).

## 4  Summary and Conclusion

Our results suggest that decision making in the primate brain may be governed by the dual principles of Bayesian inference and reward maximization as implemented within the framework of partially observable Markov decision processes (POMDPs). The model provides a unified explanation for experimental data previously explained by two competing models, namely, the additive offset model and the dynamic weighting model for incorporating prior knowledge. In particular, the model predicts psychometric and chronometric data for the random dots motion discrimination task [13] as well as Carpenter and Williams' saccadic eye movement task [10].

Previous models of decision making, such as the LATER model [10] and the drift diffusion model [25, 15], have provided *descriptive* accounts of reaction time and accuracy data but often require assumptions such as a collapsing bound, urgency signal, or dynamic weighting to fully explain the data [26, 21, 22, 13]. Our model provides a *normative* account of the data, illustrating how the subject's choices can be interpreted as being optimal under the framework of POMDPs.

Our model relies on the principle of reward maximization to explain how an animal's decisions are influenced by changes in prior probability. The same principle also allows us to predict how an animal's choice is influenced by changes in the reward function. Specifically, the model predicts that the optimal policy $\pi^*$ is determined by the ratio $\frac{R_N - R_P}{R_S}$ and the prior probability $\Pr(d_R)$. Thus, a testable prediction of the model is that the speed-accuracy trade-off in tasks such as the random dots task is governed by the ratio $\frac{R_N - R_P}{R_S}$: smaller penalties for sampling ($R_S$) will increase accuracy and reaction time, as will larger rewards for correct choices ($R_P$) or greater penalties for errors ($R_N$). Since the reward parameters in our model represent internal reward, our model also provides a bridge to study the relationship between physical reward and subjective reward.

In our model of the random dots discrimination task, belief is expressed in terms of a piecewise normal distribution with the domain of the hidden variable $x \in (-\infty, \infty)$. A piecewise beta distribution with domain $x \in [-1, 1]$ fits the experimental data equally well. However, the beta distribution's conjugate prior is the multinomial, which can limit the application of this model. For example, the observations in the Carpenter and Williams' model cannot easily be described by a discrete value. The belief in our model can be expressed by any distribution, even a non-parametric one, as long as the observation model provides a faithful representation of the stimuli and captures the essential relationship between the stimuli and the hidden world state.

The POMDP model provides a unifying framework for a variety of perceptual decision making tasks. Our state variable $x$ and action variable $a$ work with arbitrary state and action spaces, ranging from multiple alternative choices to high dimensional real value choices. The state variables can also be dynamic, with $x_t$ following a Markov chain. Currently, we have assumed that the stimuli are independent from one time step to the next, but most real world stimuli are temporally correlated. Our model is suitable for decision tasks with time-varying state and observations that are time dependent within a trial (as long as they are conditional independent given the time-varying hidden state sequence). We thus expect our model to be applicable to significantly more complicated tasks than the ones modeled here.

## Footnotes

[1] In the decision making tasks that we model in this paper, the hidden state is fixed within a trial and thus there is no transition distribution to include in the belief update equation. However, the POMDP framework is entirely valid for time-varying states.

[2]Explicitly, $\Pr(s|b_t, A_S) = Z_t^{-1}\mathcal{N}(s|\mu_t, \sigma_e^2 + \sigma_t^2)[\Pr(d_R) + (1 - 2\Pr(d_R))\Phi(0 \mid \frac{\frac{\mu_t}{\sigma_t^2} + \frac{s}{\sigma_e^2}}{\frac{1}{\sigma_t^2} + \frac{1}{\sigma_e^2}}, (\frac{1}{\sigma_t^2} + \frac{1}{\sigma_e^2})^{-1})])$.

# References

[1] D. Knill and W. Richards. *Perception as Bayesian inference*. Cambridge University Press, 1996.

[2] R.S. Zemel, P. Dayan, and A. Pouget. Probabilistic interpretation of population codes. *Neural Computation*, 10(2), 1998.

[3] R.P.N. Rao. Bayesian computation in recurrent neural circuits. *Neural Computation*, 16(1):1–38, 2004.

[4] W.J. Ma, J.M. Beck, P.E. Latham, and A. Pouget. Bayesian inference with probabilistic population codes. *Nature Neuroscience*, 9(11):1432–1438, 2006.

[5] N.D. Daw, A.C. Courville, and D.S.Touretzky. Representation and timing in theories of the dopamine system. *Neural Computation*, 18(7):1637–1677, 2006.

[6] P. Dayan and N.D. Daw. Decision theory, reinforcement learning, and the brain. *Cognitive, Affective and Behavioral Neuroscience*, 8:429–453, 2008.

[7] R. Bogacz and T. Larsen. Integration of reinforcement learning and optimal decision making theories of the basal ganglia. *Neural Computation*, 23:817–851, 2011.

[8] C.T. Law and J. I. Gold. Reinforcement learning can account for associative and perceptual learning on a visual-decision task. *Nat. Neurosci*, 12(5):655–663, 2009.

[9] J. Drugowitsch, and A. K. Churchland R. Moreno-Bote, M. N. Shadlen, and A. Pouget. The cost of accumulating evidence in perceptual decision making. *J. Neurosci*, 32(11):3612–3628, 2012.

[10] R.H.S. Carpenter and M.L.L. Williams. Neural computation of log likelihood in the control of saccadic eye movements. *Nature*, 377:59–62, 1995.

[11] M.C. Dorris and D.P. Munoz. Saccadic probability influences motor preparation signals and time to saccadic initiation. *J. Neurosci*, 18:7015–7026, 1998.

[12] J.I. Gold, C.T. Law, P. Connolly, and S. Bennur. The relative influences of priors and sensory evidence on an oculomotor decision variable during perceptual learning. *J. Neurophysiol*, 100(5):2653–2668, 2008.

[13] T.D. Hanks, M.E. Mazurek, R. Kiani, E. Hopp, and M.N. Shadlen. Elapsed decision time affects the weighting of prior probability in a perceptual decision task. *Journal of Neuroscience*, 31(17):6339–6352, 2011.

[14] J.D. Roitman and M.N. Shadlen. Response of neurons in the lateral intraparietal area during a combined visual discrimination reaction time task. *Jounral of Neuroscience*, 22, 2002.

[15] R. Bogacz, E. Brown, J. Moehlis, P. Hu, P. Holmes, and J.D. Cohen. The physics of optimal decision making: A formal analysis of models of performance in two-alternative forced choice tasks. *Psychological Review*, 113:700–765, 2006.

[16] R. Ratcliff and G. McKoon. The diffusion decision model: Theory and data for two-choice decision tasks. *Neural Computation*, 20:127–140, 2008.

[17] P. L. Frazier and A. J. Yu. Sequential hypothesis testing under stochastic deadlines. *In Advances in Neural Information procession Systems*, 20, 2007.

[18] R.P.N. Rao. Decision making under uncertainty: A neural model based on POMDPs. *Frontiers in Computational Neuroscience*, 4(146), 2010.

[19] L. P. Kaelbling, M. L. Littman, and A. R. Cassandra. Planning and acting in partially observable stochastic domains. *Artificial Intelligence*, 101:99–134, 1998.

[20] R.S. Sutton and A.G. Barto. *Reinforcement Learning: An Introduction*. The MIT Press, 1998.

[21] P.E. Latham, Y. Roudi, M. Ahmadi, and A. Pouget. Deciding when to decide. *Soc. Neurosci. Abstracts*, 740(10), 2007.

[22] A. K. Churchland, R. Kiani, and M. N. Shadlen. Decision-making with multiple alternatives. *Nat. Neurosci.*, 11(6), 2008.

[23] R.D. Luce. *Response times: their role in inferring elementary mental organization*. Oxford University Press, 1986.

[24] M.E. Mazurek, J.D. Roitman, J. Ditterich, and M.N. Shadlen. A role for neural integrators in perceptual decision-making. *Cerebral Cortex*, 13:1257–1269, 2003.

[25] J. Palmer, A.C. Huk, and M.N. Shadlen. The effects of stimulus strength on the speed and accuracy of a perceptual decision. *Journal of Vision*, 5:376–404, 2005.

[26] J. Ditterich. Stochastic models and decisions about motion direction: Behavior and physiology. *Neural Networks*, 19:981–1012, 2006.

